# Adaptive Sparseness Using Jeffreys Prior

**Mário A. T. Figueiredo**
Institute of Telecommunications,
and Department of Electrical and Computer Engineering.
Instituto Superior Técnico
1049-001 Lisboa, Portugal
*mtf @lx.it.pt*

## Abstract

In this paper we introduce a new sparseness inducing prior which does not involve any (hyper)parameters that need to be adjusted or estimated. Although other applications are possible, we focus here on supervised learning problems: regression and classification. Experiments with several publicly available benchmark data sets show that the proposed approach yields state-of-the-art performance. In particular, our method outperforms support vector machines and performs competitively with the best alternative techniques, both in terms of error rates and sparseness, although it involves no tuning or adjusting of sparseness-controlling hyper-parameters.

## 1 Introduction

The goal of supervised learning is to infer a functional relation $y = f(\mathbf{x})$, based on a set of (maybe *noisy*) training examples $\mathcal{D} = \{(\mathbf{x}_1, y_1), ..., (\mathbf{x}_n, y_n)\}$. Usually, the inputs are vectors, $\mathbf{x}_i = [x_{i,1}, ..., x_{i,d}]^T \in I\!\!R^d$. When $y$ is continuous (typically $y \in I\!\!R$), we are in the context of *regression*, whereas in *classification* $y$ is of categorical nature (*e.g.*, $y \in \{-1, 1\}$). Usually, the structure of $f(\cdot)$ is assumed fixed and the objective is to estimate a vector of parameters $\boldsymbol{\beta}$ defining it; accordingly we write $y = f(\mathbf{x}, \boldsymbol{\beta})$.

To achieve good *generalization* (*i.e.* to perform well on yet unseen data) it is necessary to control the *complexity* of the learned function (see [1] - [4], and the many references therein). In Bayesian approaches, complexity is controlled by placing a prior on the function to be learned, *i.e.*, on $\boldsymbol{\beta}$. This should not be confused with a *generative* (*informative*) Bayesian approach, since it involves no explicit modelling of the joint probability $p(\mathbf{x}, y)$. A common choice is a zero-mean Gaussian prior, which appears under different names, like *ridge regression* [5], or *weight decay*, in the neural learning literature [6]. Gaussian priors are also used in non-parametric contexts, like the Gaussian processes (GP) approach [2], [7], [8], [9], which has roots in earlier spline models [10] and regularized radial basis functions [11]. Very good performance has been reported for methods based on Gaussian priors [8], [9]. Their main disadvantage is that they do not control the structural complexity of the resulting functions. That is, if one of the components of $\boldsymbol{\beta}$ (say, a weight in a neural network) happens to be irrelevant, a Gaussian prior will not set it exactly to zero, thus

---

This work was partially supported by the Portuguese Foundation for Science and Technology (FCT), Ministry of Science and Technology, under project POSI/33143/SRI/2000.

pruning that parameter, but to some small value.

Sparse estimates (*i.e.*, in which irrelevant parameters are set exactly to zero) are desirable because (in addition to other learning-theoretic reasons [4]) they correspond to a structural simplification of the estimated function. Using Laplacian priors (equivalently, $l_1$-penalized regularization) is known to promote sparseness [12] - [15]. *Support vector machines* (SVM) take a non-Bayesian approach to the goal of sparseness [2], [4]. Interestingly, however, it can be shown that the SVM and $l_1$-penalized regression are closely related [13].

Both in approaches based on Laplacian priors and in SVMs, there are hyper-parameters which control the degree of sparseness of the obtained estimates. These are commonly adjusted using cross-validation methods which do not optimally utilize the available data, and are time consuming. We propose an alternative approach which involves no hyper-parameters. The key steps of our proposal are: (i) a hierarchical Bayes interpretation of the Laplacian prior as a *normal/independent* distribution (as used in robust regression [16]); (ii) a Jeffreys' non-informative second-level hyper-prior (in the same spirit as [17]) which expresses scale-invariance and, more importantly, is parameter-free [18]; (iii) a simple *expectation-maximization* (EM) algorithm which yields a *maximum a posteriori* (MAP) estimate of $\boldsymbol{\beta}$ (and of the observation noise variance, in the case of regression).

Our method is related to the *automatic relevance determination* (ARD) concept [7], [19], which underlies the recently proposed *relevance vector machine* (RVM) [20], [21]. The RVM exhibits state-of-the-art performance, beating SVMs both in terms of accuracy and sparseness [20], [21]. However, we do not resort to a *type-II maximum likelihood* approximation [18] (as in ARD and RVM); rather, our modelling assumptions lead to a marginal *a posteriori* probability function on $\boldsymbol{\beta}$ whose mode is located by a very simple EM algorithm. Like the RVM, but unlike the SVM, our classifier produces probabilistic outputs.

Experimental evaluation of the proposed method, both with synthetic and real data, shows that it performs competitively with (often better than) GP-based methods, RVM, and SVM.

## 2   Regression

We consider functions of the type $f(\mathbf{x}, \boldsymbol{\beta}) = \boldsymbol{\beta}^T \mathbf{h}(\mathbf{x})$, *i.e.*, that are linear with respect to $\boldsymbol{\beta}$ (whose dimensionality we will denote by $k$). This includes: (i) classical linear regression, $\mathbf{h}(\mathbf{x}) = [1, x_1, ..., x_d]^T$; (ii) nonlinear regression via a set of $k$ basis functions, $\mathbf{h}(\mathbf{x}) = [\phi_1(\mathbf{x}), ..., \phi_k(\mathbf{x})]^T$; (iii) kernel regression, $\mathbf{h}(\mathbf{x}) = [1, K(\mathbf{x}, \mathbf{x}_1), ..., K(\mathbf{x}, \mathbf{x}_n)]^T$, where $K(\mathbf{x}, \mathbf{y})$ is some (symmetric) kernel function [2] (as in SVM and RVM regression), not necessarily verifying Mercer's condition.

We follow the standard assumption that $y_i = f(\mathbf{x}_i, \boldsymbol{\beta}) + w_i$, for $i = 1, ..., n$, where $[w_1, ..., w_n]$ is a set of independent zero-mean Gaussian variables with variance $\sigma^2$. With $\mathbf{y} \equiv [y_1, ..., y_n]^T$, the likelihood function is then $p(\mathbf{y}|\boldsymbol{\beta}) = \mathcal{N}(\mathbf{y}|\mathbf{H}\boldsymbol{\beta}, \sigma^2\mathbf{I})$, where $\mathbf{H}$ is the $(n \times k)$ *design matrix* which depends on the $\mathbf{x}_i$s and on the adopted function representation, and $\mathcal{N}(\mathbf{v}|\boldsymbol{\mu}, \mathbf{C})$ denotes a Gaussian density of mean $\boldsymbol{\mu}$ and covariance $\mathbf{C}$, evaluated at $\mathbf{v}$.

With a zero-mean Gaussian prior with covariance $\mathbf{A}$, $p(\boldsymbol{\beta}|\mathbf{A}) = \mathcal{N}(\boldsymbol{\beta}|0, \mathbf{A})$, the posterior $p(\boldsymbol{\beta}|\mathbf{y})$ is still Gaussian with mean and mode at

$$\widehat{\boldsymbol{\beta}} = (\sigma^2 \mathbf{A}^{-1} + \mathbf{H}^T\mathbf{H})^{-1}\mathbf{H}^T\mathbf{y}.$$

When $\mathbf{A}$ is proportional to identity, say $\mathbf{A} = \mu^2\mathbf{I}$, this is called *ridge regression* [5].

With a Laplacian prior for $\boldsymbol{\beta}$, $p(\boldsymbol{\beta}|\alpha) = \prod_i p(\beta_i|\alpha)$, with $p(\beta_i|\alpha) = \frac{\alpha}{2}\exp\{-\alpha|\beta_i|\}$, the posterior $p(\boldsymbol{\beta}|\mathbf{y})$ is not Gaussian. The *maximum a posteriori* (MAP) estimate is given by

$$\widehat{\boldsymbol{\beta}} = \arg\min\{\|\mathbf{H}\boldsymbol{\beta} - \mathbf{y}\|_2^2 + 2\sigma^2\alpha\|\boldsymbol{\beta}\|_1\}, \tag{1}$$

where $\|\mathbf{v}\|_2$ is the Euclidean ($l_2$) norm, and $\|\mathbf{v}\|_1 = \sum_i |v_i|$ is the $l_1$ norm. In linear regression this is called the LASSO (*least absolute shrinkage and selection operator*) [14]. The main effect of the $l_1$ penalty is that some of the components of $\widehat{\boldsymbol{\beta}}$ may be exactly zero. If $\mathbf{H}$ is an orthogonal matrix, (1) can be solved separately for each $\beta_i$, leading to the *soft threshold* estimation rule, widely used in wavelet-based signal/image denoising [22].

Let us consider an alternative model: let each $\beta_i$ have a zero-mean Gaussian prior $p(\beta_i|\tau_i) = \mathcal{N}(\beta_i|0, \tau_i)$, with its own variance $\tau_i$ (like in ARD and RVM). Now, rather than adopting a *type-II maximum likelihood* criterion (as in ARD and RVM), let us consider hyper-priors for the $\tau_i$s and integrate them out. Assuming exponential hyper-priors $p(\tau_i|\gamma) = (\gamma/2) \exp\{-\gamma\, \tau_i/2\}$ (for $\tau_i \geq 0$, because these are variances) we obtain

$$p(\beta_i|\gamma) = \int_0^\infty p(\beta_i|\tau_i)p(\tau_i|\gamma)\, d\tau_i = \frac{\sqrt{\gamma}}{2} \exp\{-\sqrt{\gamma}\, |\beta_i|\}.$$

This shows that the Laplacian prior is equivalent to a 2-level hierachical-Bayes model: zero-mean Gaussian priors with independent exponentially distributed variances. This decomposition has been exploited in robust *least absolute deviation* (LAD) regression [16].

The hierarchical decomposition of the Laplacian prior allows using the EM algorithm to implement the LASSO criterion in (1) by simply regarding $\boldsymbol{\tau} = [\tau_1, ..., \tau_k]$ as *hidden/missing data*. In fact, the complete log-posterior (with a flat prior for $\sigma^2$, and where $\boldsymbol{\Upsilon}(\boldsymbol{\tau}) \equiv \mathrm{diag}(\tau_1^{-1}, ..., \tau_m^{-1})$),

$$\log p(\boldsymbol{\beta}, \sigma^2|\mathbf{y}, \boldsymbol{\tau}) \propto -n \log \sigma^2 - \frac{\|\mathbf{y} - \mathbf{H}\boldsymbol{\beta}\|_2^2}{\sigma^2} - \boldsymbol{\beta}^T \boldsymbol{\Upsilon}(\boldsymbol{\tau})\boldsymbol{\beta}, \qquad (2)$$

is easy to maximize with respect to $\boldsymbol{\beta}$ and $\sigma^2$. The E-step reduces to the computation of the conditional expectation of $\boldsymbol{\Upsilon}(\boldsymbol{\tau})$, given current (at iteration $t$) estimates $\widehat{\sigma^2}_{(t)}$ and $\widehat{\boldsymbol{\beta}}_{(t)}$. This leads to $\mathbf{V}_{(t)} \equiv E[\boldsymbol{\Upsilon}(\boldsymbol{\tau})|\mathbf{y}, \widehat{\sigma^2}_{(t)}, \widehat{\boldsymbol{\beta}}_{(t)}] = \gamma\, \mathrm{diag}(|\widehat{\beta}_{1,(t)}|^{-1}, ..., |\widehat{\beta}_{k,(t)}|^{-1})$. The M-step is then defined by the two following update equations:

$$\widehat{\sigma^2}_{(t+1)} = \frac{1}{n}\|\mathbf{y} - \mathbf{H}\widehat{\boldsymbol{\beta}}_{(t)}\|_2^2 \qquad (3)$$

and

$$\widehat{\boldsymbol{\beta}}_{(t+1)} = (\widehat{\sigma^2}_{(t+1)}\mathbf{V}_{(t)} + \mathbf{H}^T\mathbf{H})^{-1}\mathbf{H}^T\mathbf{y}. \qquad (4)$$

This EM algorithm is not the most efficient way to solve (1); see, *e.g.*, the methods proposed in [23], [14]. Our main goal is to open the way to the adoption of different hyper-priors.

One question remains: how to adjust $\gamma$, which controls the degree of sparseness of the estimates? Our proposal is to remove $\gamma$ from the model, by replacing the exponential hyper-prior by a non-informative Jeffreys hyper-prior: $p(\tau_i) \propto \tau_i^{-1}$. This prior expresses ignorance with respect to scale (see [17], [18]) and, most importantly, it is parameter-free. Of course this is no longer equivalent to a Laplacian prior on $\boldsymbol{\beta}$, but to some other prior. As will be shown experimentally, this prior strongly induces sparseness and yields state-of-the-art performance. Computationally, this choice leads to a minor modification of the EM algorithm described above: matrix $\mathbf{V}_{(t)}$ is now given by $\mathbf{V}_{(t)} = \mathrm{diag}(|\widehat{\beta}_{1,(t)}|^{-2}, ..., |\widehat{\beta}_{k,(t)}|^{-2})$.

Since several of the $\widehat{\beta}_i$s may go to zero, it is not convenient to deal with $\mathbf{V}_{(t)}$. However, we can re-write the M-step as

$$\widehat{\boldsymbol{\beta}}_{(t+1)} = \mathbf{U}_{(t)}(\widehat{\sigma^2}_{(t+1)}\mathbf{I} + \mathbf{U}_{(t)}\mathbf{H}^T\mathbf{H}\mathbf{U}_{(t)})^{-1}\mathbf{U}_{(t)}\mathbf{H}^T\mathbf{y},$$

where $\mathbf{U}_{(t)} \equiv \mathrm{diag}(|\widehat{\beta}_{1,(t)}|, ..., |\widehat{\beta}_{k,(t)}|)$, thus avoiding the inversion of the elements of $\widehat{\boldsymbol{\beta}}_{(t)}$. Moreover, it is not necessary to invert the matrix, but simply to solve the corresponding linear system, whose dimension is only the number of non-zero elements in $\mathbf{U}_{(t)}$.

## 3 Regression experiments

Our first example illustrates the use of the proposed method for variable selection in standard linear regression. Consider a sequence of 20 true $\beta$s, having from 1 to 20 non-zero components (out of 20): from $[3, 0, 0, ..., 0]$ to $[3, 3, ..., 3]$. For each $\beta$, we obtain 100 random ($50 \times 20$) design matrices, following the procedure in [14], and for each of these, we obtain data points with unit noise variance. Fig. 1 (a) shows the mean number of estimated non-zero components, as a function of the true number. Our method exhibits a very good ability to find the correct number of nonzero components in $\beta$, in an adaptive manner.

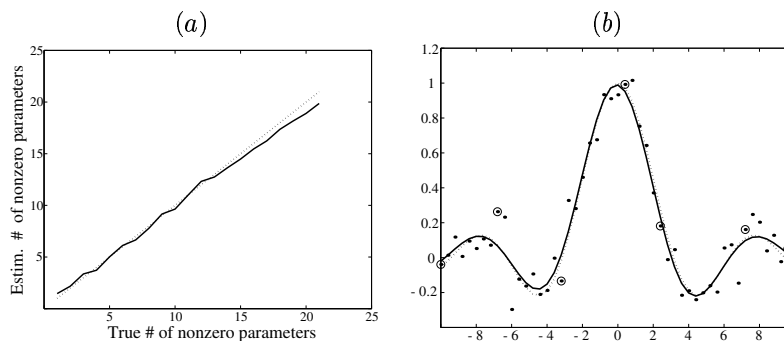

Figure 1: (a) Mean number of nonzero components in $\widehat{\beta}$ versus the number of nonzero components in $\beta$ (the dotted line is the identity). (b) Kernel regression. Dotted line: true function $y = \sin(x)/x$. Dots: 50 noisy observations ($\sigma = 0.1$). Solid line: the estimated function. Circles: data points corresponding to the non-zero parameters.

We now consider two of the experimental setups of [14]: $\beta_a = [3, 1.5, 0, 0, 2, 0, 0, 0]$, with $\sigma = 3$, and $\beta_b = [5, 0, 0, 0, 0, 0, 0, 0]$, with $\sigma = 2$. In both cases, $n = 20$, and the design matrices are generated as in [14]. In table 3, we compare the relative modelling error ($ME = E[\|\mathbf{H}\widehat{\beta} - \mathbf{H}\beta\|^2]$) improvement (with respect to the least squares solution) of our method and of several methods studied in [14]. Our method performs comparably with the best method for each case, although it involves no tuning or adjustment of parameters, and is computationally faster.

Table 1: Relative (%) improvement in modeling error of several mehods.

| Method | $\beta_a$ | $\beta_b$ |
|---|---|---|
| Proposed method | 28% | 74% |
| LASSO (CV) | 13% | 69% |
| LASSO (GCV) | 30% | 65% |
| Subset selection | 13% | 77% |

We now study the performance of our method in kernel regression, using Gaussian kernels, i.e., $K(\mathbf{x}, \mathbf{x}_i) = \exp\{-\|\mathbf{x} - \mathbf{x}_i\|^2/(2h^2)\}$. We begin by considering the synthetic example studied in [20] and [21], where the true function is $y = \sin(x)/x$ (see Fig. 1 (b)). To compare our results to the RVM and the variational RVM (VRVM), we ran the algorithm on 25 generations of the noisy data. The results are summarized in Table 2 (which also includes the SVM results from [20]). Finally, we have also applied our method to the well-known *Boston housing* data-set (20 random partitions of the full data-set into 481 training samples and 25 test samples); Table 2 shows the results, again versus SVM, RVM, and VRVM regression (as reported in [20]). In these tests, our method performs better than

RVM, VRVM, and SVM regression, although it doesn't require any tuning.

Table 2: Mean root squared errors and mean number of kernels for the "$\sin(x)/x$" function and the Boston housing examples.

| "$\sin(x)/x$" function | | | | Boston housing | | |
|---|---|---|---|---|---|---|
| Method | MSE | No. kernels | | Method | MSE | No. kernels |
| New method | 0.0455 | 7.0 | | New method | 9.98 | 45.2 |
| SVM | 0.0519 | 28.0 | | SVM | 10.29 | 235.2 |
| RVM | 0.0494 | 6.9 | | RVM | 10.17 | 41.1 |
| VRVM | 0.0494 | 7.4 | | VRVM | 10.36 | 40.9 |

## 4 Classification

In classification the formulation is somewhat more complicated, with the standard approach being *generalized linear models* [24]. For a two-class problem ($y \in \{-1, 1\}$), the probability that an observation $\mathbf{x}$ belongs to, say class 1, is given by a nonlinear function $\psi : \mathbb{R} \to [0, 1]$ (called the link), $P(y = 1|\mathbf{x}) = \psi(\boldsymbol{\beta}^T \mathbf{h}(\mathbf{x}))$, where $\mathbf{h}(\mathbf{x})$ can have one of the forms referred in the first paragraph of Section 2 (linear, nonlinear, kernel).

Although the most common choice for $\psi$ is the *logistic* function, $\psi(z) = (1 + \exp(-z))^{-1}$, in this paper, we adopt the probit model $\psi(z) = \Phi(z)$, where

$$\Phi(z) \equiv \int_{-\infty}^{z} \mathcal{N}(x|0, 1) \, dx, \tag{5}$$

the standard Gaussian cumulative distribution function (cdf). The probit model has a simple interpretation in terms of hidden variables [25], which we will exploit. Consider a hidden variable $z = \boldsymbol{\beta}^T \mathbf{h}(\mathbf{x}) + w$, where $p(w) = \mathcal{N}(w|0, 1)$. Then, if the classification rule is $y = 1$ if $z \geq 0$, and $y = -1$ if $z < 0$, we obtain the probit model:

$$P(y = 1|\mathbf{x}) = P(\boldsymbol{\beta}^T \mathbf{h}(\mathbf{x}) + w \geq 0) = \Phi(\boldsymbol{\beta}^T \mathbf{h}(\mathbf{x})).$$

Given training data $\mathcal{D} = \{(\mathbf{x}_1, y_1), ..., (\mathbf{x}_n, y_n)\}$, consider the corresponding vector of hidden/missing variables $\mathbf{z} = [z_1, ..., z_n]^T$. If we had $\mathbf{z}$, we would have a simple linear regression likelihood $p(\mathbf{z}|\boldsymbol{\beta}) = \mathcal{N}(\mathbf{z}|\mathbf{H}\boldsymbol{\beta}, \mathbf{I})$. This fact suggests using the EM algorithm to estimate $\boldsymbol{\beta}$, by treating $\mathbf{z}$ as missing data.

To promote sparseness, we will adopt the same hierarchical prior on $\boldsymbol{\beta}$ that we have used for regression: $p(\beta_i|\tau_i) = \mathcal{N}(\beta_i|0, \tau_i)$ and $p(\tau_i) \propto 1/\tau_i$ (the Jeffreys prior). The complete log posterior (with the hidden vectors $\boldsymbol{\tau}$ and $\mathbf{z}$) is

$$\log p(\boldsymbol{\beta}|\mathbf{y}, \boldsymbol{\tau}, \mathbf{z}) \propto -\boldsymbol{\beta}^T \mathbf{H}^T \mathbf{H} \boldsymbol{\beta} - 2\boldsymbol{\beta}^T \mathbf{H}^T \mathbf{z} - \boldsymbol{\beta}^T \boldsymbol{\Upsilon}(\boldsymbol{\tau})\boldsymbol{\beta}, \tag{6}$$

which is similar to (2), except for the noise variance which is not needed here, and for the fact that now $\mathbf{z}$ is missing. The expected value of $\boldsymbol{\Upsilon}(\boldsymbol{\tau})$ is similar to the regression case; accordingly we define the same diagonal matrix $\mathbf{U}_{(t)} = \text{diag}(|\widehat{\beta}_{1,(t)}|, ..., |\widehat{\beta}_{k,(t)}|)$. In addition, we also need $E[\mathbf{z}|\widehat{\boldsymbol{\beta}}_{(t)}, \mathbf{y}]$ (notice that the complete log-posterior is linear with respect to $\mathbf{z}$), which can be expressed in closed form, for each $z_i$, as

$$s_{i,(t)} \equiv E[z_i|\widehat{\boldsymbol{\beta}}_{(t)}, \mathbf{y}] = \begin{cases} \widehat{\boldsymbol{\beta}}_{(t)}^T \mathbf{h}(\mathbf{x}_i) + \dfrac{\mathcal{N}(\widehat{\boldsymbol{\beta}}_{(t)}^T \mathbf{h}(\mathbf{x}_i)|0, 1)}{1 - \Phi(-\widehat{\boldsymbol{\beta}}_{(t)}^T \mathbf{h}(\mathbf{x}_i))} & \text{if } y_i = 1 \\[4mm] \widehat{\boldsymbol{\beta}}_{(t)}^T \mathbf{h}(\mathbf{x}_i) - \dfrac{\mathcal{N}(\widehat{\boldsymbol{\beta}}_{(t)}^T \mathbf{h}(\mathbf{x}_i)|0, 1)}{\Phi(-\widehat{\boldsymbol{\beta}}_{(t)}^T \mathbf{h}(\mathbf{x}_i))} & \text{if } y_i = -1. \end{cases} \tag{7}$$

These expressions are easily derived after noticing that $z_i$ is (conditionally) Gaussian with mean $\widehat{\boldsymbol{\beta}}_{(t)}^T \mathbf{h}(\mathbf{x}_i)$, but left-truncated at zero if $y_i = 1$, and right-truncated at zero if $y_i = -1$. With $\mathbf{s}_{(t)} \equiv [s_{1,(t)}, ..., s_{n,(t)}]^T$, the M-step is similar to the regression case,

$$\widehat{\boldsymbol{\beta}}_{(t+1)} = \mathbf{U}_{(t)}(\mathbf{I} + \mathbf{U}_{(t)}\mathbf{H}^T\mathbf{H}\mathbf{U}_{(t)})^{-1}\mathbf{U}_{(t)}\mathbf{H}^T\mathbf{s}_{(t)},$$

with $\mathbf{s}_{(t)}$ playing the role of observed data.

## 5  Classification experiments

In all the experiments we use kernel classifiers, with Gaussian kernels, *i.e.*, $K(\mathbf{x}, \mathbf{x}_i) = \exp\{-\|\mathbf{x} - \mathbf{x}_i\|^2/(2\,h^2)\}$, where $h$ is a parameter that controls the kernel width.

Our first experiment is mainly illustrative and uses Ripley's synthetic data[1]; the optimal error rate for this problems is $8\%$ [3]. Table 3 shows the average test set error (on 1000 test samples) and the final number of kernels, for 20 classifiers learned from 20 random subsets of size 100 from the original 250 training samples. For comparison, we also include results (from [20]) for RVM, variational RVM (VRVM), and SVM classifiers. On this data set, our method performs competitively with RVM and VRVM and much better than SVM (specially in terms of sparseness). To allow the comparisons, we chose $h = 0.5$, as in [20].

Table 3 also reports the numbers of errors achieved by the proposed method and by several state-of-the-art techniques on three well-known benchmark problems: the *Pima Indians diabetes*[2], the *Leptograpsus crabs*[2], and the *Wisconsin breast cancer* [3] (*WBC*). For the WBC, we report average results over 30 random partitions (300/269 training/testing, as in [26]). All the inputs are normalized to zero mean and unit variance, and the kernel width was set to $h = 4$, for the *Pima* and *crabs* problems, and to $h = 12$ for the WBC. On the *Pima* and *crabs* data sets, our algorithm outperforms all the other techniques. On the WBC data set, our method performs nearly as well as the best available alternative. The running time of our learning algorithm (in MATLAB, on a PIII-800MHz) is less than 1 second for *crabs*, and about 2 seconds for the *Pima* and WBC problems. Finally, notice that the classifiers obtained with our algorithm are much sparser than the SVM classifiers.

Table 3: Numbers of test set errors for the four data sets studied (see text for details). The numbers in square brackets in the "method" column indicate the bibliographic reference from which the results are quoted. The numbers in parentheses indicate the (mean) number of kernels used by the classifiers (when available).

| Method | Ripley's | Pima | Crabs | WBC |
|---|---|---|---|---|
| Proposed method | 94 (4.8) | 61 (6) | 0 (5) | 8.5 (5) |
| SVM [20] | 106 (38) | 64 (110) | N/A | N/A |
| RVM [20] | 93 (4) | 65 (4) | N/A | N/A |
| VRVM [20] | 92 (4) | 65 (4) | N/A | N/A |
| SVM [26] | N/A | 64 | 4 | 9 |
| Neural network [9] | N/A | 75 | 3 | N/A |
| Logistic regression [9] | N/A | 66 | 4 | N/A |
| Linear discriminant [26] | N/A | 67 | 3 | 19 |
| Gaussian process [9], [26] | N/A | 68, 67 | 3 | 8 |

## 6 Concluding remarks

We have introduced a new sparseness inducing prior related to the Laplacian prior. Its main feature is the absence of any hyper-parameters to be adjusted or estimated. Experiments with several publicly available benchmark data sets, both for regression and classification, have shown state-of-the-art performance. In particular, our approach outperforms support vector machines and Gaussian process classifiers both in terms of error rate and sparseness, although it involves no tuning or adjusting of sparseness-controlling hyper-parameters.

Future research includes testing on large-scale problems, like handwritten digit classification. One of the weak points of our approach, when used with kernel-based methods, is the need to solve a linear system in the M-step (of dimension equal to the number of training points) whose computational requirements make it impractical to use with very large training data sets. This issue is of current interest to researchers in kernel-based methods (*e.g.*, [27]), and we also intend to focus on it.

## Footnotes

[1] Available (divided into training/test sets) at: http://www.stats.ox.ac.uk/pub/PRNN/

[2] Available at http://www.stats.ox.ac.uk/pub/PRNN/

[3] Available at: http://www.ics.uci.edu/ mlearn/MLSummary.html

## References

[1] V. Cherkassky and F. Mulier, *Learning from Data: Concepts, Theory, and Methods*. New York: Wiley, 1998.

[2] N. Cristianini and J. Shawe-Taylor, *Support Vector Machines and Other Kernel-Based Learning Methods*. Cambridge University Press, 2000.

[3] B. Ripley, *Pattern Recognition and Neural Networks*. Cambridge University Press, 1996.

[4] V. Vapnik, *Statistical Learning Theory*. New York: John Wiley, 1998.

[5] A. Hoerl and R. Kennard, "Ridge regression: Biased estimation for nonorthogonal problems," *Technometrics*, vol. 12, pp. 55–67, 1970.

[6] C. Bishop, *Neural Networks for Pattern Recognition*. Oxford University Press, 1995.

[7] R. Neal, *Bayesian Learning for Neural Networks*. New York: Springer Verlag, 1996.

[8] C. Williams, "Prediction with Gaussian processes: from linear regression to linear prediction and beyond," in *Learning and Inference in Graphical Models*, Kluwer, 1998.

[9] C. Williams and D. Barber, "Bayesian classification with Gaussian priors," *IEEE Trans. on Pattern Analysis and Machine Intelligence*, vol. 20, no. 12, pp. 1342–1351, 1998.

[10] G. Kimeldorf and G. Wahba, "A correspondence between Bayesian estimation of stochastic processes and smoothing by splines," *Annals of Mathematical Statistics*, vol. 41, pp. 495–502, 1990.

[11] T. Poggio and F. Girosi, "Networks for approximation and learning," *Proceedings of the IEEE*, vol. 78, pp. 1481–1497, 1990.

[12] S. Chen, D. Donoho, and M. Saunders, "Atomic decomposition by basis pursuit," *SIAM Journal of Scientific Computation*, vol. 20, no. 1, pp. 33–61, 1998.

[13] F. Girosi, "An equivalence between sparse approximation and support vector machines," *Neural Computation*, vol. 10, pp. 1445–1480, 1998.

[14] R. Tibshirani, "Regression shrinkage and selection via the lasso," *Journal of the Royal Statistical Society (B)*, vol. 58, 1996.

[15] P. Williams, "Bayesian regularization and pruning using a Laplace prior," *Neural Computation*, vol. 7, pp. 117–143, 1995.

[16] K. Lange and J. Sinsheimer, "Normal/independent distributions and their applications in robust regression," *Journal of Computational and Graphical Statistics*, vol. 2, pp. 175–198, 1993.

[17] M. Figueiredo and R. Nowak, "Wavelet-based image estimation: an empirical Bayes approach using Jeffreys' noninformative prior," *IEEE Transactions on Image Processing*, vol. 10, pp. 1322-1331, 2001.

[18] J. Berger, *Statistical Decision Theory and Bayesian Analysis.* Springer-Verlag, 1980.

[19] D. MacKay, "Bayesian non-linear modelling for the 1993 energy prediction competition," in *Maximum Entropy and Bayesian Methods*, G. Heidbreder, ed., pp. 221–234, Kluwer, 1996.

[20] C. Bishop and M. Tipping, "Variational relevance vector machines," in *Proceedings of the 16th Conference in Uncertainty in Artificial Intelligence*, pp. 46–53, Morgan Kaufmann, 2000.

[21] M. Tipping, "The relevance vector machine," in *Advances in Neural Information Processing Systems – NIPS 12* (S. Solla, T. Leen, and K.-R. Müller, eds.), pp. 652–658, MIT Press, 2000.

[22] D. L. Donoho and I. M. Johnstone, "Ideal adaptation via wavelet shrinkage," *Biometrika*, vol. 81, pp. 425–455, 1994.

[23] M. Osborne, B. Presnell, and B. Turlach, "A new approach to variable selection in least squares problems," *IMA Journal of Numerical Analysis*, vol. 20, pp. 389–404, 2000.

[24] P. McCullagh and J. Nelder, *Generalized Linear Models*. London: Chapman and Hall, 1989.

[25] J. Albert and S. Chib, "Bayesian analysis of binary and polychotomous response data," *Journal of the American Statistical Association*, vol. 88, pp. 669–679, 1993.

[26] M. Seeger, "Bayesian model selection for support vector machines, Gaussian processes and other kernel classifiers," in *Advances in Neural Information Processing – NIPS 12* (S. Solla, T. Leen, and K.-R. Müller, eds.), pp. 603–609, MIT Press, 2000.

[27] C. Williams and M. Seeger, "Using the Nystrom method to speedup kernel machines," in *NIPS 13*, MIT Press, 2001.
